# Generalized Regularized Least-Squares Learning with Predefined Features in a Hilbert Space

**Wenye Li, Kin-Hong Lee, Kwong-Sak Leung**
Department of Computer Science and Engineering
The Chinese University of Hong Kong
Shatin, Hong Kong, China
{wyli, khlee, ksleung}@cse.cuhk.edu.hk

## Abstract

Kernel-based regularized learning seeks a model in a hypothesis space by minimizing the empirical error and the model's complexity. Based on the representer theorem, the solution consists of a linear combination of translates of a kernel. This paper investigates a generalized form of representer theorem for kernel-based learning. After mapping predefined features and translates of a kernel simultaneously onto a hypothesis space by a specific way of constructing kernels, we proposed a new algorithm by utilizing a generalized regularizer which leaves part of the space unregularized. Using a squared-loss function in calculating the empirical error, a simple convex solution is obtained which combines predefined features with translates of the kernel. Empirical evaluations have confirmed the effectiveness of the algorithm for supervised learning tasks.

## 1   Introduction

Supervised learning, or learning from examples, refers to the task of training a system by a set of examples which are specified by input-output pairs. The system is used to predict the output value for any valid input object after training. Examples of such tasks include regression which produces continuous values, and classification which predicts a class label for an input object.

Vapnik's seminal work[1] shows that the key to effectively solving this problem is by controlling the solution's complexity, which leads to the techniques known as regularized kernel methods[1] [2][3] and regularization networks[4]. The work championed by Poggio and other researchers[5][6] implicitly treats learning as an approximation problem and gives a general scheme with ideas going back to modern regularization theory[7][8][9]. For both frameworks, a solution is sought by simultaneously minimizing the empirical error and the complexity. More precisely, given a training set $D = (\mathbf{x}_i; y_i)_{i=1}^m$, an estimator $f : \mathcal{X} \to \mathcal{Y}$, where $\mathcal{X}$ is a closed subset of $\mathcal{R}^d$ and $\mathcal{Y} \subset \mathcal{R}$, is given by

$$\min_{f \in \mathcal{H}_K} \frac{1}{m} \sum_{i=1}^m V\left(y_i, f\left(\mathbf{x}_i\right)\right) + \gamma \left\|f\right\|_K^2 \tag{1}$$

where $V$ is a convex loss function, $\|f\|_K$ is the norm of $f$ in a reproducing kernel Hilbert space (RKHS) $\mathcal{H}_K$ induced by a positive definite function (a kernel) $K_{\mathbf{x}}\left(\mathbf{x}'\right) = K\left(\mathbf{x}, \mathbf{x}'\right)$, and $\gamma$ is a regularization parameter that makes a trade-off between the empirical error and the complexity. $\gamma \left\|f\right\|_K^2$ is also called a regularizer.

According to representer theorem [10][11] [12], the minimizer of (1) admits a simple solution as a linear combination of translates of the kernel $K$ by the training data

$$f^* = \sum_{i=1}^m c_i K_{\mathbf{x}_i}, \; c_i \in \mathcal{R}, 1 \leq i \leq m$$

for a variety of loss functions. Different loss functions lead to different learning algorithms. For example, when used for classification, a squared-loss $(y - f(\mathbf{x}))^2$ brings about the regularized least-squares classification (RLSC) algorithm[13][14][15]; while a hinge loss $(1 - yf(\mathbf{x}))_+ \equiv \max(1 - yf(\mathbf{x}), 0)$ corresponds to the classical support vector machines(SVM).

Using this model, data are implicitly projected onto the hypothesis space $\mathcal{H}_K$ via a transformation

$$\phi_K : \mathbf{x} \rightarrow K_{\mathbf{x}}$$

and a linear functional is sought by finding its representer in $\mathcal{H}_K$, which generally has infinite dimensions. It is generally believed that learning problems associated with infinite dimensions are ill-posed and need regularization. However, finite dimensional problems are often associated with well-posedness and do not need regularization. Motivated by this, we unified these two views in this paper. Using an existing trick in designing kernels, an RKHS is constructed which contains a subspace spanned by some predefined features and this subspace is left unregularized during the learning process. Empirical results have shown the embedding of these features often has the effect of *stabilizing* the algorithms's performance for different choices of kernels and prevents the results from deteriorating for inappropriate kernels.

The paper is organized as follows. First, a generalized regularized learning model and its associated representer theorem are studied. Then, we introduce an existing trick with which we constructed a hypothesis space which has a subspace of the predefined features. Next, a generic learning algorithm is proposed based on the model and especially evaluated for classification problems. Empirical results have confirmed the benefits brought by the algorithm.

A note on notation. Throughout the paper, vectors and matrices are represented in bold notation and scalars in normal script, e.g. $\mathbf{x}_1, \cdots, \mathbf{x}_m \in \mathcal{R}^d$, $\mathbf{K} \in \mathcal{R}^{m \times m}$, and $y_1, \cdots, y_m \in \mathcal{R}$. $\mathbf{I}$ and $\mathbf{O}$ are used to denote an identity matrix and a zero matrix of appropriate sizes, respectively. For clarity, the size of a matrix is sometimes added as a subscript, such as $\mathbf{O}_{m \times \ell}$.

## 2 Generalized regularized least-squares learning model

Suppose the space $\mathcal{H}_K$ decomposes into the direct sum: $\mathcal{H}_K = \mathcal{H}_0 \oplus \mathcal{H}_1$, where $\mathcal{H}_0$ is spanned by $\ell (\leq m)$ linearly independent features: $\mathcal{H}_0 = span(\varphi_1, \cdots, \varphi_\ell)$. We propose the generalized regularized least-squares (G-RLS) learning model as

$$\min_{f \in \mathcal{H}_K} L(f) = \frac{1}{m} \sum_{i=1}^{m} (y_i - f(\mathbf{x}_i))^2 + \gamma \|f - Pf\|_K^2, \tag{2}$$

where $Pf$ is the orthogonal projection of $f$ onto $\mathcal{H}_0$.

Suppose $f^*$ is the minimizer of (2). For any $f \in \mathcal{H}_K$, let $f = f^* + \delta g$ where $\delta \in \mathcal{R}$ and $g \in \mathcal{H}_K$. Now take derivative w.r.t. $\delta$ and notice that $\frac{\partial L}{\partial \delta}|_{\delta=0} = 0$. Then

$$-\frac{2}{m} \sum_{i=1}^{m} (y_i - f^*(\mathbf{x}_i)) g(\mathbf{x}_i) + 2\gamma \langle f^* - Pf^*, g \rangle_K = 0, \tag{3}$$

where $\langle \cdot, \cdot \rangle_K$ denotes the inner product in $\mathcal{H}_K$. This equation holds for any $g \in \mathcal{H}_K$. In particular, setting $g = K_{\mathbf{x}}$ gives

$$f^* - Pf^* = \frac{\sum_{i=1}^{m} (y_i - f^*(\mathbf{x}_i)) K_{\mathbf{x}_i}}{m\gamma}. \tag{4}$$

$Pf^*$ is the orthogonal projection of $f^*$ onto $\mathcal{H}_0$ and hence,

$$Pf^* = \sum_{p=1}^{\ell} \lambda_p \varphi_p, \ \lambda_p \in \mathcal{R}, 1 \leq p \leq \ell. \tag{5}$$

So (4) is simplified to

$$f^* = \sum_{p=1}^{\ell} \lambda_p \varphi_p + \sum_{i=1}^{m} c_i K_{\mathbf{x}_i}, \tag{6}$$

where

$$c_i = \frac{y_i - f^*(\mathbf{x}_i)}{m\gamma}, \ 1 \leq i \leq m. \tag{7}$$

The coefficients $\lambda_1, \cdots, \lambda_\ell, c_1, \cdots, c_m$ are uniquely specified by $m + \ell$ linear equations. The first $m$ equations are obtained by substituting (6) into (7). The rest $\ell$ equations are derived from the orthogonality constraint between $Pf^*$ and $f - Pf^*$, which can be written as

$$\left\langle \varphi_p, \sum_{i=1}^{m} c_i K_{\mathbf{x}_i} \right\rangle_K = 0, \ 1 \leq p \leq \ell, \tag{8}$$

or equivalently due to the property of reproducing kernels,

$$\sum_{i=1}^{m} c_i \varphi_p(\mathbf{x}_i) = 0, \ 1 \leq p \leq \ell. \tag{9}$$

The solution (6) derived from (2) satisfies the *reproduction* property. Suppose $(\mathbf{x}_i; y_i)_{i=1}^{m}$ comes purely from a model which is perfectly linearly related to $\varphi_1, \cdots, \varphi_\ell$, it is desirable to get back a solution that is independent of the other features. As an evident result of (2), the property is satisfied. The parameters $c_1, \cdots, c_m$ in the resulting estimator (6) are all zero, which makes the regularizer in (2) equal to zero.

## 3 Kernel construction

By decomposing a hypothesis space $\mathcal{H}_K$ and studying a generalized regularizer, we have proposed the G-RLS model and derived a solution which consists of predefined features as well as translates of a kernel function. In this section, starting with predefined features $\varphi_1, \cdots, \varphi_\ell$ and a kernel $\Phi$, we will construct a hypothesis space which contains the features and translates of the kernel by using an existing trick.

### 3.1 A kernel construction trick

Let's consider the following reproducing kernel

$$K(\mathbf{x}, \mathbf{x}') = H(\mathbf{x}, \mathbf{x}') + \sum_{p=1}^{\ell} \varphi'_p(\mathbf{x}) \varphi'_p(\mathbf{x}') \tag{10}$$

where

$$\begin{aligned} H(\mathbf{x}, \mathbf{x}') &= \Phi(\mathbf{x}, \mathbf{x}') - \sum_{p=1}^{\ell} \varphi'_p(\mathbf{x}) \Phi(\mathbf{x}_p, \mathbf{x}') - \sum_{q=1}^{\ell} \varphi'_q(\mathbf{x}') \Phi(\mathbf{x}, \mathbf{x}_q) \\ &\quad + \sum_{p=1}^{\ell} \sum_{q=1}^{\ell} \varphi'_p(\mathbf{x}) \varphi'_q(\mathbf{x}') \Phi(\mathbf{x}_p, \mathbf{x}_q), \end{aligned} \tag{11}$$

$\Phi$ is any strictly positive definite function, and $\varphi'_1, \cdots, \varphi'_\ell$ defines a linear transformation of $\varphi_1, \cdots, \varphi_\ell$ w.r.t. $\mathbf{x}_1, \cdots, \mathbf{x}_\ell$,

$$\begin{bmatrix} \varphi'_1(\mathbf{x}) \\ \cdots \\ \varphi'_\ell(\mathbf{x}) \end{bmatrix} = \begin{bmatrix} \varphi_1(\mathbf{x}_1) & \cdots & \varphi_1(\mathbf{x}_\ell) \\ \cdots & & \cdots \\ \varphi_\ell(\mathbf{x}_1) & \cdots & \varphi_\ell(\mathbf{x}_\ell) \end{bmatrix}^{-1} \begin{bmatrix} \varphi_1(\mathbf{x}) \\ \cdots \\ \varphi_\ell(\mathbf{x}) \end{bmatrix} \tag{12}$$

which satisfies

$$\varphi'_q(\mathbf{x}_p) = \begin{cases} 1 & 1 \leq p = q \leq \ell \\ 0 & 1 \leq p \neq q \leq \ell \end{cases}. \tag{13}$$

This trick is studied in [16] to provide an alternative basis for radial basis functions and first used in a fast RBF interpolation algorithm[17]. A sketch of properties which are peripheral to our concerns in this paper are given below.

$$K_{\mathbf{x}_p} = \varphi'_p, \ 1 \leq p \leq \ell \tag{14}$$

$$\langle \varphi'_p, \varphi'_q \rangle_K = \begin{cases} 1 & 1 \le p = q \le \ell \\ 0 & 1 \le p \ne q \le \ell \end{cases} \tag{15}$$

$$H_{\mathbf{x}_p} = H(\mathbf{x}_p, \cdot) = 0, \ 1 \le p \le \ell \tag{16}$$

$$\langle H_{\mathbf{x}_i}, \varphi'_p \rangle_K = \langle H(\mathbf{x}_i, \cdot), \varphi'_p \rangle_K = 0, \ \ell+1 \le i \le m, 1 \le p \le \ell \tag{17}$$

$$\langle H_{\mathbf{x}_i}, H_{\mathbf{x}_j} \rangle_K = H(\mathbf{x}_i, \mathbf{x}_j), \ \ell+1 \le i, j \le m \tag{18}$$

Another property is that the matrix $\mathbf{H} = (H(\mathbf{x}_i, \mathbf{x}_j))_{i,j=\ell+1}^m$ is strictly positive definite, which will be used in the computations below.

By constructing a kernel $K$ using this trick, predefined features $\varphi_1, \cdots, \varphi_\ell$ are explicitly mapped onto $\mathcal{H}_K$ which has a subspace $\mathcal{H}_0 = span(\varphi'_1, \cdots, \varphi'_\ell) = span(\varphi_1, \cdots, \varphi_\ell)$. By property (15), we can see that $\varphi'_1, \cdots, \varphi'_\ell$ also forms an orthonormal basis of $\mathcal{H}_0$.

### 3.2 Computation

After projecting the features $\varphi_1, \cdots, \varphi_\ell$ onto an RKHS $\mathcal{H}_K$, let's study the regularized minimization problem in (2). As shown in (6), the minimizer has a form of a linear combination of predefined features and translates of a kernel. By the properties of $K$ in (14)-(17), the minimizer can be rewritten as:

$$
\begin{aligned}
f^* &= \sum_{p=1}^\ell \lambda_p \varphi_p + \sum_{i=1}^m c_i K_{\mathbf{x}_i} \\
&= \sum_{p=1}^\ell \lambda'_p \varphi'_p + \left( \sum_{i=1}^\ell c_i \varphi'_i + \sum_{i=\ell+1}^m c_i \left( H_{\mathbf{x}_i} + \sum_{p=1}^\ell \varphi'_p(\mathbf{x}_i) \varphi'_p \right) \right) \\
&= \sum_{p=1}^\ell \left( \lambda'_p + c_p + \sum_{i=\ell+1}^m c_i \varphi'_p(\mathbf{x}_i) \right) \varphi'_p + \sum_{i=\ell+1}^m c_i H_{\mathbf{x}_i} \\
&= \sum_{p=1}^\ell \tilde{\lambda}_p \varphi'_p + \sum_{i=\ell+1}^m \tilde{c}_i H_{\mathbf{x}_i}
\end{aligned}
\tag{19}
$$

where $\tilde{\lambda}_1, \cdots, \tilde{\lambda}_\ell, \tilde{c}_{\ell+1}, \cdots, \tilde{c}_m$ are $m$ parameters to be determined. Furthermore, from the orthogonal property between $\varphi'_p$ and $H_{\mathbf{x}_i}$ in (17), we have

$$f^* - Pf^* = \sum_{i=\ell+1}^m \tilde{c}_i H_{\mathbf{x}_i}. \tag{20}$$

To determine the values of $\tilde{\lambda} = \left( \tilde{\lambda}_1, \cdots, \tilde{\lambda}_\ell \right)^T$ and $\tilde{\mathbf{c}} = (\tilde{c}_{\ell+1}, \cdots, \tilde{c}_m)^T$, we need

$$\|f^* - Pf^*\|_K^2 = \sum_{i,j=\ell+1}^m \tilde{c}_i \tilde{c}_j H(\mathbf{x}_i, \mathbf{x}_j) = \begin{pmatrix} \tilde{\lambda} \\ \tilde{\mathbf{c}} \end{pmatrix}^T \tilde{\mathbf{H}} \begin{pmatrix} \tilde{\lambda} \\ \tilde{\mathbf{c}} \end{pmatrix} \tag{21}$$

where $\tilde{\mathbf{H}} = \begin{pmatrix} \mathbf{O}_{\ell \times \ell} & \mathbf{O}_{\ell \times (m-\ell)} \\ \mathbf{O}_{(m-\ell) \times \ell} & \mathbf{H} \end{pmatrix}$. Substituting (21) into (2), we have

$$L = \frac{1}{m} \left( \mathbf{y} - \tilde{\mathbf{K}} \begin{pmatrix} \tilde{\lambda} \\ \tilde{\mathbf{c}} \end{pmatrix} \right)^T \left( \mathbf{y} - \tilde{\mathbf{K}} \begin{pmatrix} \tilde{\lambda} \\ \tilde{\mathbf{c}} \end{pmatrix} \right) + \gamma \begin{pmatrix} \tilde{\lambda} \\ \tilde{\mathbf{c}} \end{pmatrix}^T \tilde{\mathbf{H}} \begin{pmatrix} \tilde{\lambda} \\ \tilde{\mathbf{c}} \end{pmatrix} \tag{22}$$

where $\tilde{\mathbf{K}} = \begin{pmatrix} \mathbf{I}_{\ell \times \ell} & \mathbf{O}_{\ell \times (m-\ell)} \\ \mathbf{E}^T & \mathbf{H} \end{pmatrix}$ and $\mathbf{E} = (\varphi'_p(\mathbf{x}_i))_{p=1, i=\ell+1}^{\ell, m}$. Take derivative w.r.t. $\begin{pmatrix} \tilde{\lambda} \\ \tilde{\mathbf{c}} \end{pmatrix}$ and set the derivative to zero, and we get

$$\tilde{\mathbf{K}}^2 \begin{pmatrix} \tilde{\lambda} \\ \tilde{\mathbf{c}} \end{pmatrix} + \gamma m \tilde{\mathbf{H}} \begin{pmatrix} \tilde{\lambda} \\ \tilde{\mathbf{c}} \end{pmatrix} = \tilde{\mathbf{K}} \mathbf{y}. \tag{23}$$

Since $\tilde{\mathbf{K}}^{-1} = \begin{pmatrix} \mathbf{I}_{\ell\times\ell} & \mathbf{O}_{(m-\ell)\times\ell} \\ -\mathbf{H}^{-1}\mathbf{E}^T & \mathbf{H}^{-1} \end{pmatrix}$ and $\tilde{\mathbf{K}}^{-1}\tilde{\mathbf{H}} = \tilde{\mathbf{I}} = \begin{pmatrix} \mathbf{O}_{\ell\times\ell} & \mathbf{O}_{\ell\times(m-\ell)} \\ \mathbf{O}_{(m-\ell)\times\ell} & \mathbf{I}_{(m-\ell)\times(m-\ell)} \end{pmatrix}$,

we have

$$\left(\tilde{\mathbf{K}} + \gamma m\tilde{\mathbf{I}}\right)\begin{pmatrix} \tilde{\lambda} \\ \tilde{\mathbf{c}} \end{pmatrix} = \mathbf{y}, \tag{24}$$

i.e.

$$\begin{pmatrix} \mathbf{I}_{\ell\times\ell} & \mathbf{O}_{\ell\times(m-\ell)} \\ \mathbf{E}^T & \mathbf{H} + \gamma m\mathbf{I} \end{pmatrix}\begin{pmatrix} \tilde{\lambda} \\ \tilde{\mathbf{c}} \end{pmatrix} = \begin{pmatrix} \mathbf{y}_1 \\ \mathbf{y}_2 \end{pmatrix}, \tag{25}$$

where $\mathbf{y}_1 = (y_1, \cdots, y_\ell)^T$ and $\mathbf{y}_2 = (y_{\ell+1}, \cdots, y_m)^T$. Equation (25) uniquely specifies $\tilde{\lambda}$ by

$$\tilde{\lambda} = \mathbf{y}_1, \tag{26}$$

and $\tilde{\mathbf{c}}$ by

$$(\mathbf{H} + \gamma m\mathbf{I})\tilde{\mathbf{c}} = \mathbf{y}_2 - \mathbf{E}^T\tilde{\lambda}. \tag{27}$$

$\mathbf{H} + \gamma m\mathbf{I}$ is a strictly positive definite matrix. The equation can be efficiently solved either by conjugate gradient or by Cholesky factorization. The worst case complexity is $O\left((m-\ell)^3\right) \approx O\left(m^3\right)$. It is also possible to investigate iterative methods for solving linear systems coupled with recent advances in fast matrix-vector multiplication methods (e.g. fast multipole method), and the complexity reduces to nearly $O\left(m\log m\right)$, which provides the potential to solve large scale problems.

## 4 A generic learning algorithm

Based on the discussions above, a generic learning algorithm (G-RLS algorithm) is summarized below.

1. Start with data $(\mathbf{x}_i; y_i)_{i=1}^m$.
2. For $\ell\,(\leq m)$ predefined linearly independent features $\varphi_1, \cdots, \varphi_\ell$ of the data, define $\varphi_1', \cdots, \varphi_\ell'$ according to equation (12).
3. Choose a symmetric, strictly positive definite function $\Phi_{\mathbf{x}}(\mathbf{x}') = \Phi(\mathbf{x}, \mathbf{x}')$ which is continuous on $\mathcal{X} \times \mathcal{X}$. Define $H$ according to equation (11).
4. The estimator $f : \mathcal{X} \to \mathcal{Y}$ is given by

$$f(\mathbf{x}) = \sum_{p=1}^\ell \tilde{\lambda}_p \varphi_p'(\mathbf{x}) + \sum_{i=\ell+1}^m \tilde{c}_i H_{\mathbf{x}_i}(\mathbf{x}) \tag{28}$$

where $\tilde{\lambda}_1, \cdots, \tilde{\lambda}_\ell, \tilde{c}_{\ell+1}, \cdots, \tilde{c}_m$ are obtained by solving equations (26) and (27).

The algorithm can be applied to a number of applications including regression and binary classification. As a simple example for regression, noisy points were randomly generated via a function $y = |5 - x|$, and we fitted the data by a curve. Polynomial features up to the second degree ($\varphi_1 = 1, \varphi_2 = x, \varphi_3 = x^2$) were used for G-RLS algorithm along with a Gaussian RBF kernel

$$\Phi_x(\cdot) = e^{-\frac{\|x - \cdot\|^2}{\sigma^2}}.$$

We selected ridge regression with the Gaussian RBF kernel for a comparison, which can be regarded as an implementation of standard regularized least-squares model for regression tasks. For both algorithms, three trials were made in which the parameter $\sigma$ was set to a large value, to a small value, and by cross validation respectively. For each $\sigma$, the parameter $\gamma$ was set by cross validation. Comparing with ridge regression in figure 1(b), the existence of polynomial features in G-RLS has the effect of *stabilizing* the results, as shown in figure 1(a). Varying $\sigma$, different fitting results were obtained by ridge regression. However, for G-RLS algorithm, the difference was not evident.

In the case of generalized regularized least-squares classification (G-RLSC), each $y_i$ of the training set takes the values $\{-1, 1\}$. The predicted label of any $\mathbf{x}$ depends on the sign of (28)

$$y = \begin{cases} 1, & f(\mathbf{x}) > 0 \\ -1 & otherwise \end{cases}.$$

G-RLSC uses the "classical" squared-loss as a classification loss criterion. The effectiveness of this criterion has been reported by the empirical results[13][14][15].

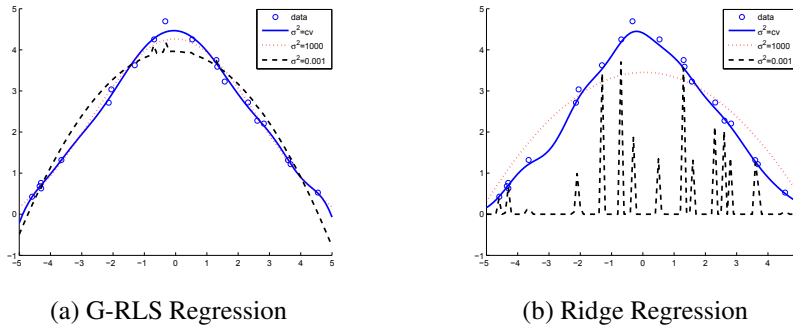

| (a) G-RLS Regression | (b) Ridge Regression |

Figure 1: A Regression Example. The existence of polynomial features in G-RLS helped to improve the *stability* of the algorithm.

## 5 Experiments

To evaluate the performance of G-RLS algorithm, empirical results are reported on text categorization tasks using the three datasets from CMU text mining group[1]. The *7-sectors* dataset has $4,573$ web pages belonging to seven economic sectors, with each sector containing pages varying from $300$ to $1,099$. The *4-universities* dataset consists of $8,282$ webpages collected mainly from four universities, in which the pages belong to seven classes and each class has $137$ to $3,764$ pages. The *20-newsgroups* dataset collects UseNet postings into twenty newsgroups and each group has about $1,000$ messages. We experimented with its four major subsets. The first subset has $5$ groups (comp.*), the second $4$ groups (rec.*), the third $4$ groups (sci.*) and the last $4$ groups (talk.*).

For each dataset, we removed all but the $2,000$ words with highest mutual information with the class variable by rainbow package[18]. The document was represented as *bag-of-words* with linear normalization into $[-1, 1]$. Probabilistic latent semantic analysis[19] (pLSA) was used to get ten latent features $\varphi_1, \cdots, \varphi_{10}$ out of the data. Experiments were carried out with different number ($100\tilde{\;}3,200$) of data for training and the rest for testing. Each experiment consisted of ten runs and the average accuracy is reported. In each run, the data were separated by the *xval-prep* utility accompanied in C4.5 package[2].

Figure 2 compares the performance of G-RLSC, RLSC and SVM. It is shown that G-RLSC reports improved results on most of the datasets except on *4-universities*. Moreover, an insightful observation may find that although SVM excels on the dataset when the number of training data increases, G-RLSC shows better performance than standard RLSC. A possible reason is that the hinge loss used by SVM is more appropriate than the squared-loss used by RLSC and G-RLSC on this dataset; while the embedding of pLSA features still improves the accuracy.

## 6 Conclusion

In this paper, we first proposed a generic G-RLS learning model. Unlike the standard kernel-based methods which only consider the translates of a kernel for model learning, the new model takes predefined features into special consideration. A generalized regularizer is studied which leaves part of the hypothesis space unregularized. Similar ideas were explored in spline smoothing[9] in which low degree polynomials are not regularized. Another example is semi-parametric SVM[2], which considers the addition of some features to the kernel expansion for SVM. However, to our knowledge, few learning algorithms and applications have been studied along this line from a unified RKHS regularization point of view, or investigated for empirical evaluations.

The second part of our work presented a practical computation method based on the model. An RKHS that contains the combined solutions is explicitly constructed based on a special trick in designing kernels. (The idea of a conditionally positive definite function[20] is lurking in the back-

[1]http://www.cs.cmu.edu/~TextLearning/datasets.html
[2]http://www.rulequest.com/Personal/c4.5r8.tar.gz.

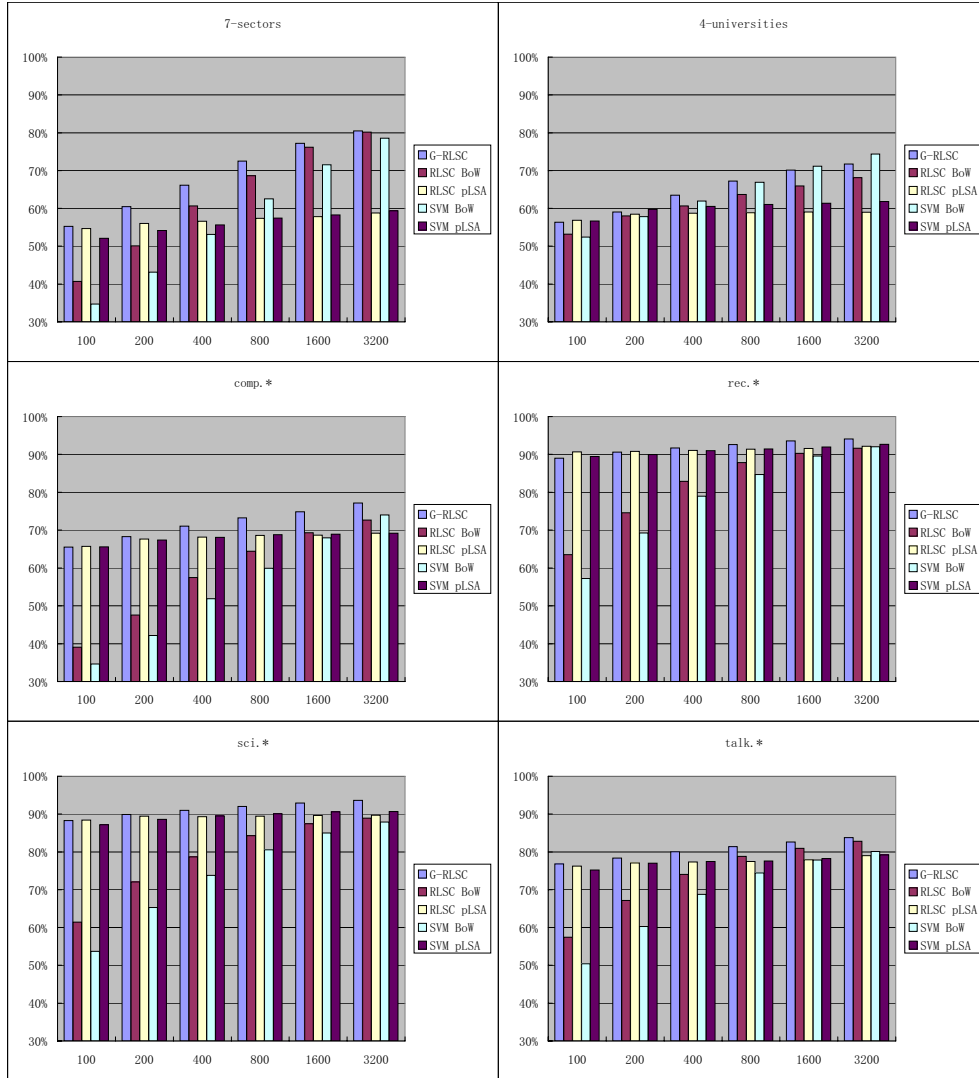

Figure 2: Classification accuracies on CMU text datasets with different number of training samples. Ten pLSA features along with a linear kernel $\Phi$ were used for G-RLSC. Both *bag-of-words* (BoW) and pLSA representations of documents were experimented for RLSC and SVM with a linear kernel. The parameter $\gamma$ was selected via cross validation. For multi-classification, G-RLSC and RLSC used one-versus-all strategy. SVM used one-versus-one strategy.

ground of this trick, which goes beyond the discussion of this paper.) With the construction of the RKHS, the computation is further optimized and the theoretical analysis of such algorithms is also potentially facilitated.

We evaluated G-RLS learning algorithm in text categorization. The empirical results from real-world applications have confirmed the effectiveness of the algorithm.

### Acknowledgments

The authors thank Dr. Haixuan Yang for useful discussions. This research was partially supported by RGC Earmarked Grant #4173/04E and #4132/05E of Hong Kong SAR and RGC Research Grant Direct Allocation of the Chinese University of Hong Kong.

# References

[1] V.N. Vapnik. *Statistical Learning Theory*. John Wiley and Sons, 1998.

[2] B. Schölkopf and A.J. Smola. *Learning with Kernels*. The MIT Press, 2002.

[3] J.S. Taylor and N. Cristianini. *Kernel Methods for Pattern Analysis*. Cambridge University Press, 2004.

[4] T. Evgeniou, M. Pontil, and T. Poggio. Regularization networks and support vector machines. *Adv. Comput. Math.*, 13:1–50, 2000.

[5] T. Poggio and F. Girosi. Regularization algorithms for learning that are equivalent to multilayer networks. *Science*, 247:978–982, 1990.

[6] T. Poggio and S. Smale. The mathematics of learning: Dealing with data. *Not. Am. Math. Soc*, 50:537–544, 2003.

[7] A.N. Tikhonov and V.Y. Arsenin. *Solutions of Ill-Posed Problems*. Winston and Sons, 1977.

[8] V.A. Morozov. *Methods for Solving Incorrectly Posed Problems*. Springer-Verlag, 1984.

[9] G. Wahba. *Spline Models for Observational Data*. SIAM, 1990.

[10] G. Kimeldorf and G. Wahba. Some results on Tchebycheffian spline functions. *J. Math. Anal. Appl.*, 33:82–95, 1971.

[11] F. Girosi, M.J. Jones, and T. Poggio. Regularization theory and neural networks architectures. *Neural Comput.*, 7:219–269, 1995.

[12] B. Schölkopf, R. Herbrich, and A.J. Smola. A generalized representer theorem. In *COLT'2001 and EuroCOLT'2001*, 2001.

[13] R.M. Rifkin. *Everything Old is New Again: A Fresh Look at Historical Approaches in Machine Learning*. PhD thesis, Massachusetts Institute of Technology, 2002.

[14] G. Fung and O.L. Mangasarian. Proximal support vector machine classifiers. In *KDD'01*, 2001.

[15] J.A.K. Suykens and J. Vandewalle. Least squares support vector machine classifiers. *Neural Process. Lett.*, 9:293–300, 1999.

[16] W. Light and H. Wayne. Spaces of distributions, interpolation by translates of a basis function and error estimates. *J. Numer. Math.*, 81:415–450, 1999.

[17] R.K. Beatson, W.A. Light, and S. Billings. Fast solution of the radial basis function interpolation equations: Domain decomposition methods. *SIAM J. Sci. Comput.*, 22:1717–1740, 2000.

[18] A.K. McCallum. Bow: A toolkit for statistical language modeling, text retrieval, classification and clustering. http://www.cs.cmu.edu/∼mccallum/bow, 1996.

[19] T. Hofmann. Probabilistic latent semantic analysis. In *UAI'99*, 1999.

[20] C.A. Micchelli. Interpolation of scattered data: Distances, matrices, and conditionally positive definite functions. *Constr. Approx.*, 2:11–22, 1986.
